# Construction of Nonparametric Bayesian Models from Parametric Bayes Equations

**Peter Orbanz**
University of Cambridge and ETH Zurich
`p.orbanz@eng.cam.ac.uk`

## Abstract

We consider the general problem of constructing nonparametric Bayesian models on infinite-dimensional random objects, such as functions, infinite graphs or infinite permutations. The problem has generated much interest in machine learning, where it is treated heuristically, but has not been studied in full generality in nonparametric Bayesian statistics, which tends to focus on models over probability distributions. Our approach applies a standard tool of stochastic process theory, the construction of stochastic processes from their finite-dimensional marginal distributions. The main contribution of the paper is a generalization of the classic Kolmogorov extension theorem to conditional probabilities. This extension allows a rigorous construction of nonparametric Bayesian models from systems of finite-dimensional, parametric Bayes equations. Using this approach, we show (i) how existence of a conjugate posterior for the nonparametric model can be guaranteed by choosing conjugate finite-dimensional models in the construction, (ii) how the mapping to the posterior parameters of the nonparametric model can be explicitly determined, and (iii) that the construction of conjugate models in essence requires the finite-dimensional models to be in the exponential family. As an application of our constructive framework, we derive a model on infinite permutations, the nonparametric Bayesian analogue of a model recently proposed for the analysis of rank data.

## 1 Introduction

Nonparametric Bayesian models are now widely used in machine learning. Common models, in particular the Gaussian process (GP) and the Dirichlet process (DP), were originally imported from statistics, but the nonparametric Bayesian idea has since been adapted to the needs of machine learning. As a result, the scope of Bayesian nonparametrics has expanded significantly: Whereas traditional nonparametric Bayesian statistics mostly focuses on models on probability distributions, machine learning researchers are interested in a variety of infinite-dimensional objects, such as functions, kernels, or infinite graphs. Initially, existing DP and GP approaches were modified and combined to derive new models, including the Infinite Hidden Markov Model [2] or the Hierarchical Dirichlet Process [15]. More recently, novel stochastic process models have been defined from scratch, such as the Indian Buffet Process (IBP) [8] and the Mondrian Process [13]. This paper studies the construction of new nonparametric Bayesian models from finite-dimensional distributions: To construct a model on a given type of infinite-dimensional object (for example, an infinite graph), we start out from available probability models on the finite-dimensional counterparts (probability models on finite graphs), and translate them into a model on infinite-dimensional objects using methods of stochastic process theory. We then ask whether interesting statistical properties of the finite-dimensional models used in the constructions, such as conjugacy of priors and posteriors, carry over to the stochastic process model.

In general, the term *nonparametric Bayesian model* refers to a Bayesian model on an infinite-dimensional parameter space. Unlike *parametric* models, for which the number of parameters is constantly bounded w.r.t. sample size, *nonparametric* models allow the number of parameters to grow with the number of observations. To accommodate a variable and asymptotically unbounded number of parameters within a single parameter space, the dimension of the space has to be infinite, and nonparametric models can be defined as statistical models with infinite-dimensional parameter spaces [17]. For a given sample of finite size, the model will typically select a finite subset of the available parameters to explain the observations. A Bayesian nonparametric model places a prior distribution on the infinite-dimensional parameter space.

Many nonparametric Bayesian models are defined in terms of their finite-dimensional marginals: For example, the Gaussian process and Dirichlet process are characterized by the fact that their finite-dimensional marginals are, respectively, Gaussian and Dirichlet distributions [11, 5]. The probability-theoretic construction result underlying such definitions is the Kolmogorov extension theorem [1], described in Sec. 2 below. In stochastic process theory, the theorem is used to study the properties of a process in terms of its marginals, and hence by studying the properties of finite-dimensional distributions. Can the statistical properties of a nonparametric Bayesian model, i.e. of a parameterized family of distributions, be treated in a similar manner, by considering the model's marginals? For example, can a nonparametric Bayesian model be guaranteed to be conjugate if the marginals used in its construction are conjugate? Techniques such as the Kolmogorov theorem construct individual distributions, whereas statistical properties are properties of parameterized families of distributions. In Bayesian estimation, such families take the form of conditional probabilities. The treatment of the statistical properties of nonparametric Bayesian models in terms of finite-dimensional Bayes equations therefore requires an extension result similar to the Kolmogorov theorem that is applicable to conditional distributions. The main contribution of this paper is to provide such a result.

We present an analogue of the Kolmogorov theorem for conditional probabilities, which permits the direct construction of conditional stochastic process models on countable-dimensional spaces from finite-dimensional conditional probabilities. Application to conjugate models shows how a conjugate nonparametric Bayesian model can be constructed from conjugate finite-dimensional Bayes equations – including the mapping to the posterior parameters. The converse is also true: To construct a conjugate nonparametric Bayesian model, the finite-dimensional models used in the construction all have to be conjugate. The construction of stochastic process models from exponential family marginals is almost generic: The model is completely described by the mapping to the posterior parameters, which has a generic form as a function of the infinite-dimensional counterpart of the model's sufficient statistic. We discuss how existing models fit into the framework, and derive the nonparametric Bayesian version of a model on infinite permutations suggested by [9]. By essentially providing a construction recipe for conjugate models of countable dimension, our theoretical results have clear practical implications for the derivation of novel nonparametric Bayesian models.

## 2 Formal Setup and Notation

Infinite-dimensional probability models cannot generally be described with densities and therefore require some basic notions of measure-theoretic probability. In this paper, required concepts will be measures on product spaces and abstract conditional probabilities (see e.g. [3] or [1] for general introductions). Randomness is described by means of an abstract probability space $(\Omega, \mathcal{A}, \mathbb{P})$. Here, $\Omega$ is a space of points $\omega$, which represent atomic random events, $\mathcal{A}$ is a $\sigma$-algebra of events on $\Omega$, and $\mathbb{P}$ a probability measure defined on the $\sigma$-algebra. A random variable is a measurable mapping from $\Omega$ into some space of observed values, such as $X : \Omega \to \Omega_x$. The distribution of $X$ is the image measure $P_X := X(\mathbb{P}) = \mathbb{P} \circ X^{-1}$. Roughly speaking, the events $\omega \in \Omega$ represent abstract states of nature, i.e. knowing the value of $\omega$ completely describes all probabilistic aspects of the model universe, and all random aspects are described by the probability measure $\mathbb{P}$. However, $\Omega$, $\mathcal{A}$ and $\mathbb{P}$ are never known explicitly, but rather constitute the modeling assumption that any explicitly known distribution $P_X$ is derived from one and the same probability measure $\mathbb{P}$ through some random variable $X$.

Multiple dimensions of random variables are formalized by product spaces. We will generally deal with an infinite-dimensional space such as $\Omega_x^E$, were $E$ is an infinite index set and $\Omega_x^E$ is the $E$-

fold product of $\Omega_x$ with itself. The set of finite subsets of $E$ will be denoted $\mathcal{F}(E)$, such that $\Omega_x^I$ with $I \in \mathcal{F}(E)$ is a finite-dimensional subspace of $\Omega_x^E$. Each product space $\Omega_x^I$ is equipped with the product Borel $\sigma$-algebra $\mathcal{B}_x^I$. Random variables with values on these spaces have product structure, such as $X^I = \bigotimes_{i \in I} X^{\{i\}}$. Note that this does *not* imply that the corresponding measure $P_X^I := X^I(\mathbb{P})$ is a product measure; the individual components of $X^I$ may be dependent. The elements of the infinite-dimensional product space $\Omega_x^E$ can be thought of as functions of the form $E \to \Omega_x$. For example, the space $\mathbb{R}^{\mathbb{R}}$ contains all real-valued functions on the line.

Product spaces $\Omega_x^I \subset \Omega_x^J$ of different dimensions are linked by a *projection operator* $\pi_{JI}$, which restricts a vector $x^J \in \Omega_x^J$ to $x^I$, the subset of entries of $x^J$ that are indexed by $I \subset J$. For a set $A^I \subset \Omega_x^I$, the preimage $\pi_{JI}^{-1} A^I$ under projection is called a *cylinder set* with base $A^I$. The projection operator can be applied to measures as $[\pi_{JI} P_X^J] := P_X^J \circ \pi_{JI}^{-1}$, so for an $I$-dimensional event $A^I \in \mathcal{B}_x^I$, we have $[\pi_{JI} P_X^J](A^I) = P_X^J(\pi_{JI}^{-1} A^I)$. In other words, a probability is assigned to the $I$-dimensional set $A^I$ by applying the $J$-dimensional measure $P_X^J$ to the cylinder with base $A^I$. The projection of a measure is just its *marginal*, that is, $[\pi_{JI} P_X^J]$ is the marginal of the measure $P_X^J$ on the lower-dimensional subspace $\Omega_x^I$.

We denote observation variables (data) by $X^I$, parameters by $\Theta^I$ and hyperparameters by $\Psi^I$. The corresponding measures and spaces are indexed accordingly, as $P_X, P_\Theta, \Omega_\theta$ etc. The likelihoods and posteriors that occur in Bayesian estimation are conditional probability distributions. Since densities are not generally applicable in infinite-dimensional spaces, the formulation of Bayesian models on such spaces draws on the abstract conditional probabilities of measure-theoretic probability, which are derived from Kolmogorov's implicit formulation of conditional expectations [3]. We will write e.g. $P_X(X|\Theta)$ for the conditional probability of $X$ given $\Theta$. For the reader familiar with the theory, we note that all spaces considered here are Borel spaces, such that regular versions of conditionals always exist, and we hence assume all conditionals to be regular conditional probabilities (Markov kernels). Introducing abstract conditional probabilities here is far beyond the possible scope of this paper. A reader not familiar with the theory should simply read $P_X(X|\Theta)$ as a conditional distribution, but take into account that these abstract objects are only uniquely defined almost everywhere. That is, the probability $P_X(X|\Theta = \theta)$ can be changed arbitrarily for those values of $\theta$ within some set of exceptions, provided that this set has measure zero. While not essential for understanding most of our results, this fact is the principal reason that limits the results to countable dimensions.

**Example: GP.** Assume that $P_X^E(X^E|\Theta^E)$ is to represent a Gaussian process with fixed covariance function. Then $X^E$ is function-valued, and if for example $E := \mathbb{R}_+$ and $\Omega_x := \mathbb{R}$, the product space $\Omega_x^E = \mathbb{R}^{\mathbb{R}_+}$ contains all functions $x^E$ of the form $x^E : \mathbb{R}_+ \to \mathbb{R}$. Each axis label $i \in E$ in the product space is a point on the real line, and a finite index set $I \in \mathcal{F}(E)$ is a finite collection of points $I = (i_1, \ldots, i_m)$. The projection $\pi_{EI} x^E$ of a function in $\Omega_x^E$ is then the vector $x^I := (x^E(i_1), \ldots, x^E(i_m))$ of function values at the points in $I$. The parameter variable $\Theta^E$ represents the mean function of the process, and so we would choose $\Omega_\theta^E := \Omega_x^E = \mathbb{R}^{\mathbb{R}_+}$.

**Example: DP.** If $P_X^E(X^E|\Theta^E)$ is a Dirichlet process, the variable $X^E$ takes values $x^E$ in the set of probability measures over a given domain, such as $\mathbb{R}$. A probability measure on $\mathbb{R}$ (with its Borel algebra $\mathcal{B}(\mathbb{R})$) is in particular a set function $\mathcal{B}(\mathbb{R}) \to [0, 1]$, so we could choose $E = \mathcal{B}(\mathbb{R})$ and $\Omega_x = [0, 1]$. The parameters of a Dirichlet process $\mathrm{DP}(\alpha, G_0)$ are a scalar concentration parameter $\alpha \in \mathbb{R}_+$, and a probability measure $G_0$ with the same domain as the randomly drawn measure $x^E$. The parameter space would therefore be chosen as $\mathbb{R}_+ \times [0, 1]^{\mathcal{B}(\mathbb{R})}$.

## 2.1 Construction of Stochastic Processes from their Marginals

Suppose that a family $P_X^I$ of probability measures are the finite-dimensional marginals of an infinite-dimensional measure $P_X^E$ (a "stochastic process"). Each measure $P_X^I$ lives on the finite-dimensional subspace $\Omega_x^I$ of $\Omega_x^E$. As marginals of one and the same measure, the measures must be marginals of each other as well:

$$P_X^I = P_X^J \circ \pi_{JI}^{-1} \qquad \text{whenever } I \subset J . \qquad (1)$$

Any family of probability measures satisfying (1) is called a *projective family*. The marginals of a stochastic process measure are always projective. A famous theorem by Kolmogorov states that the converse is also true: Any projective family on the finite-dimensional subspaces of an infinite-dimensional product space $\Omega_x^E$ uniquely defines a stochastic process on the space $\Omega_x^E$ [1]. The only assumption required is that the "axes" $\Omega_x$ of the product space are so-called *Polish spaces*, i.e.

topological spaces that are complete, separable and metrizable. Examples include Euclidean spaces, separable Banach or Hilbert spaces, countable discrete spaces, and countable products of spaces that are themselves Polish.

**Theorem 1** (Kolmogorov Extension Theorem). *Let $E$ be an arbitrary infinite set. Let $\Omega_x$ be a Polish space, and let $\{P_X^I | I \in \mathcal{F}(E)\}$ be a family of probability measures on the spaces $(\Omega_x^I, \mathcal{B}_x^I)$. If the family is projective, there exists a uniquely defined probability measure $P_X^E$ on $\Omega_x^E$ with the measures $P_X^I$ as its marginals.*

The infinite-dimensional measure $P_X^E$ constructed in Theorem 1 is called the *projective limit* of the family $P_X^I$. Intuitively, the theorem is a regularity result: The marginals determine the values of $P_X^E$ on a subset of events (namely on those events involving only a finite subset of the random variables, which are just the cylinder sets with finite-dimensional base). The theorem then states that a probability measure is such a regular object that knowledge of these values determines the measure completely, in a similar manner as continuous functions on the line are completely determined by their values on a countable dense subset. The statement of the Kolmogorov theorem is deceptive in its generality: It holds for *any* index set $E$, but if $E$ is not countable, the constructed measure $P_X^E$ is essentially useless – even though the theorem still holds, and the measure is still uniquely defined. The problem is that the measure $P_X^E$, as a set function, is not defined on the space $\Omega_x^E$, but on the $\sigma$-algebra $\mathcal{B}_x^E$ (the product $\sigma$-algebra on $\Omega_x^E$). If $E$ is uncountable, this $\sigma$-algebra is too coarse to resolve events of interest[1]. In particular, it does not contain the singletons (one-point sets), such that the measure $P_X^E$ is incapable of assigning a probability to an event of the form $\{X^E = x^E\}$.

# 3 Extension of Conditional and Bayesian Models

According to the Kolmogorov extension theorem, the properties of a stochastic process can be analyzed by studying its marginals. Can we, analogously, use a set of finite-dimensional Bayes equations to represent a nonparametric Bayesian model? The components of a Bayesian model are conditional distributions. Even though these conditionals are probability measures for (almost) each value of the condition variable, the Kolmogorov theorem cannot simply be applied to extend conditional models: Conditional probabilities are functions of two arguments, and have to satisfy a measurability requirement in the second argument (the condition). Application of the extension theorem to each value of the condition need not yield a proper conditional distribution on the infinite-dimensional space, as it disregards the properties of the second argument. But since the second argument takes the role of a parameter in statistical estimation, these properties determine the statistical properties of the model, such as sufficiency, identifiability, or conjugacy. In order to analyze the properties of an infinite-dimensional Bayesian model in terms of finite-dimensional marginals, we need a theorem that establishes a correspondence between the finite-dimensional and infinite-dimensional *conditional* distributions. Though a number of extension theorems based on conditional distributions is available in the literature, these results focus on the construction of sequential stochastic processes from a sequence of conditionals (see [10] for an overview). Theorem 2 below provides a result that, like the Kolmogorov theorem, is applicable on product spaces.

To formulate the result, the projector used to define the marginals has to be generalized from measures to conditionals. The natural way to do so is the following: If $P_X^J(X^J | \Theta^J)$ is a conditional probability on the product space $\Omega^J$, and $I \subset J$, define

$$[\pi_{JI} P_X^J](\,.\,|\Theta^J) := P_X^J(\pi_{JI}^{-1}\,.\,|\Theta^J)\,. \tag{2}$$

This definition is consistent with that of the projector above, in the sense that it coincides with the standard projector applied to the measure $P_X^J(\,.\,|\Theta^J = \theta^J)$ for any fixed value $\theta^J$ of the parameter. As with projective families of measures, we then define projective families of conditional probabilities.

**Definition 1** (Conditionally Projective Probability Models). Let $P_X^I(X^I | \Theta^I)$ be a family of regular conditional probabilities on product spaces $\Omega_x^I$, for all $I \in \mathcal{F}(E)$. The family will be called *conditionally projective* if $[\pi_{JI} P_X^J](\,.\,|\Theta^J) =_{\text{a.e.}} P_X^I(\,.\,|\Theta^I)$ whenever $I \subset J$.

As conditional probabilities are unique almost everywhere, the equality is only required to hold almost everywhere as well. In the jargon of abstract conditional probabilities, the definition requires

that $P_X^I(\,.\,|\Theta^I)$ is a version of the projection of $P_X^J(\,.\,|\Theta^J)$. Theorem 2 states that a conditional probability on a countably-dimensional product space is uniquely defined (up to a.e.-equivalence) by a conditionally projective family of marginals. In particular, if we can define a parametric model on each finite-dimensional space $\Omega_x^I$ for $I \in \mathcal{F}(E)$ such that these models are conditionally projective, the models determine an infinite-dimensional parametric model (a "nonparametric" model) on the overall space $\Omega_x^E$.

**Theorem 2** (Extension of Conditional Probabilities). *Let $E$ be a countable index set. Let $P_X^I(X^I|\Theta^I)$ be a family of regular conditional probabilities on the product space $\Omega_x^I$. Then if the family is conditionally projective, there exists a regular conditional probability $P_X^E(X^E|\mathcal{C}^E)$ on the infinite-dimensional space $\Omega_x^E$ with the $P_X^I(X^I|\Theta^I)$ as its conditional marginals. $P_X^E(X^E|\mathcal{C}^E)$ is measurable with respect to the $\sigma$-algebra $\mathcal{C}^E := \sigma(\cup_{I \in \mathcal{F}(E)} \sigma(\Theta^I))$. In particular, if the parameter variables satisfy $\pi_\Pi \Theta^J = \Theta^I$, then $P_X^E(X^E|\mathcal{C}^E)$ can be interpreted as the conditional probability $P_X^E(X^E|\Theta^E)$ with $\Theta^E := \bigotimes_{i \in E} \Theta^{\{i\}}$.*

*Proof Sketch*[2]. We first apply the Kolmogorov theorem separately for each setting of the parameters that makes the measures $P_X^I(X^I|\Theta^I = \theta^I)$ projective. For any given $\omega \in \Omega$ (the abstract probability space), projectiveness holds if $\theta^I = \Theta^I(\omega)$ for all $I \in \mathcal{F}(E)$. However, for any conditionally projective family, there is a set $N \subset \Omega$ of possible exceptions (for which projectiveness need not hold), due to the fact that conditional probabilities and conditional projections are only unique almost everywhere. Using the countability of the dimension set $E$, we can argue that $N$ is always a null set; the resulting set of constructed infinite-dimensional measures is still a valid candidate for a regular conditional probability. We then show that if this set of measures is assembled into a function of the parameter, it satisfies the measurability conditions of a regular conditional probability: We first use the properties of the marginals to show measurability on the subset of events which are preimages under projection of finite-dimensional events (the cylinder sets), and then use the $\pi$-$\lambda$ theorem [3] to extend measurability to all events. □

## 4 Conjugacy

The posterior of a Dirichlet process is again a Dirichlet process, and the posterior parameters can be computed as a function of the data and the prior parameters. This property is known as *conjugacy*, in analogy to conjugacy in parametric Bayesian models, and makes Dirichlet process inference tractable. Virtually all known nonparametric Bayesian models, including Gaussian processes, Pólya trees, and neutral-to-the-right processes are conjugate [16]. In the Bayesian and exponential family literature, conjugacy is often defined as "closure under sampling", i.e. for a given likelihood and a given class of priors, the posterior is again an element of the prior class [12]. This definition does *not* imply tractability of the posterior: In particular, the set of all probability measures (used as priors) is conjugate for any possible likelihood, but obviously this does not facilitate computation of the posterior. In the following, we call a prior and a likelihood of a Bayesian model conjugate if the posterior (i) is parameterized and (ii) there is a measurable mapping $T$ from the data $x$ and the prior parameter $\psi$ to the parameter $\psi' = T(x, \psi)$ which specifies the corresponding posterior. In the definition below, the conditional probability $k$ represents the parametric form of the posterior. The definition is applicable to "nonparametric" models, in which case the parameter simply becomes infinite-dimensional.

**Definition 2** (Conjugacy and Posterior Index). Let $P_X(X|\Theta)$ and $P_\Theta(\Theta|\Psi)$ be regular conditional probabilities. Let $P_\Theta(\Theta|X, \Psi)$ be the posterior of the model $P_X(X|\Theta)$ under prior $P_\Theta(\Theta|\Psi)$. Model and prior are called *conjugate* if there exists a regular conditional probability $k : \mathcal{B}_\theta \times \Omega_t \to [0, 1]$, parameterized on a measurable Polish space $(\Omega_t, \mathcal{B}_t)$, and a measurable map $T : \Omega_x \times \Omega_\psi \to \Omega_t$, such that

$$P_\Theta(A|X = x, \Psi = \psi) = k(A, T(x, \psi)) \qquad \text{for all } A \in \mathcal{B}_\theta . \tag{3}$$

The mapping $T$ is called the *posterior index* of the model.

The definition becomes trivial for $\Omega_t = \Omega_x \times \Omega_\psi$ and $T$ chosen as the identity mapping; it is meaningful if $T$ is reasonably simple to evaluate, and its complexity does not increase with sample size. Theorem 3 below shows that, under suitable conditions, the structure of the posterior index carries

over to the projective limit model: If the finite-dimensional marginals admit a tractable posterior index, then so does the projective limit model.

**Example.** (Posterior Indices in Exponential Families) Suppose that $P_{\mathrm{X}}(X|\Theta)$ is an exponential family model with sufficient statistic $S$ and density $p(x|\theta) = \exp(\langle S(x), \theta\rangle - \gamma(x) - \phi(\theta))$. Choose $P_\Theta(\Theta|\Psi)$ as the "natural conjugate prior" with parameters $\psi = (\alpha, y)$. Its density, w.r.t. a suitable measure $\nu_\Theta$ on parameter space, is of the form $q(\theta|\alpha, y) = K(\alpha, y)^{-1} \exp(\langle \theta, y\rangle - \alpha\phi(\theta))$. The posterior $P_\Theta(\Theta|X, \Psi)$ is conjugate in the sense of Def. 2, and its density is $q(\theta|\alpha + 1, y + S(x))$. The probability kernel $k$ is given by $k(A, (t_1, t_2)) := \int_A q(\theta|t_1, t_2) d\nu_\Theta(\theta)$, and the posterior index is $T(x, (\alpha, y)) := (\alpha + 1, y + S(x))$.

The main result of this section is Theorem 3, which explains how conjugacy carries over from the finite-dimensional to the infinite-dimensional case, and vice versa. Both extension theorems discussed so far require a projection condition on the measures and models involved. A similar condition is now required for the mappings $T^{\mathrm{I}}$: The preimages $T^{\mathrm{I},\text{-}1}$ of the posterior indices $T^{\mathrm{I}}$ must commute with the preimage under projection,

$$(\pi_{\mathrm{EI}} \circ T^{\mathrm{E}})^{\text{-}1} = (T^{\mathrm{I}} \circ \pi_{\mathrm{EI}})^{\text{-}1} \qquad \text{for all } I \in \mathcal{F}(E) \ . \tag{4}$$

The posterior indices of all well-known exponential family models, such as Gaussians and Dirichlets, satisfy this condition. The following theorem states that (i) stochastic process Bayesian models that are constructed from conjugate marginals are conjugate if the projection equation (4) is satisfied, and that (ii) such conjugate models can only be constructed from conjugate marginals.

**Theorem 3** (Functional Conjugacy of Projective Limit Models). *Let $E$ be a countable index set and $\Omega_x^E$ and $\Omega_\theta^E$ be Polish product spaces. Assume that there is a Bayesian model on each finite-dimensional subspace $\Omega_x^I$, such that the families of all priors, all observation models and all posteriors are conditionally projective. Let $P_\Theta^E(\Theta^E)$, $P_X^E(X^E|\Theta^E)$ and $P_\Theta^E(\Theta^E|X^E)$ denote the respective projective limits. Then $P_\Theta^E(\Theta^E|X^E)$ is a posterior for the infinite-dimensional Bayesian model defined by $P_X^E(X^E|\Theta^E)$ with prior $P_\Theta^E(\Theta^E)$, and the following holds:*

(i) *Assume that each finite-dimensional posterior $P_\Theta^I(\Theta^I|X^I)$ is conjugate w.r.t. its respective Bayesian model, with posterior index $T^I$ and probability kernel $k^I$. Then if there is a measurable mapping $T : \Omega_x^E \to \Omega_t^E$ satisfying the projection condition (4), the projective limit posterior $P_\Theta^E(\Theta^E|X^E)$ is conjugate with posterior index $T$.*

(ii) *Conversely, if the infinite-dimensional posterior $P_\Theta^E(\Theta^E|X^E)$ is conjugate with posterior index $T^E$ and probability kernel $k^E$, then each marginal posterior $P_\Theta^I(\Theta^I|X^I)$ is conjugate, with posterior index $T^I := \pi_{\mathrm{EI}} \circ T^E \circ \pi_{\mathrm{EI}}^{\text{-}I}$. The corresponding probability kernels $k^I$ are given by*

$$k^I(A^I, t^I) := k^E(\pi_{\mathrm{EI}}^{\text{-}I} A^I, t) \qquad \textit{for any } t \in \pi_{\mathrm{EI}}^{\text{-}I} t^I \ . \tag{5}$$

The theorem is not stated here in full generality, but under two simplifying assumptions: We have omitted the use of hyperparameters, such that the posterior indices depend only on the data, and all involved spaces (observation space, parameter space etc) are assumed to have the same dimension for each Bayesian model. Generalizing the theorem beyond both assumptions is technically not difficult, but the additional parameters and notation for book-keeping on dimensions reduce readability.

*Proof Sketch*[2]. *Part (i):* We define a candidate for the probability kernel $k^{\mathrm{E}}$ representing the projective limit posterior, and then verify that it makes the model conjugate when combined with the mapping $T$ given by assumption. To do so, we first construct the conditional probabilities $P_\Theta^{\mathrm{I}}(\Theta^{\mathrm{I}}|T^{\mathrm{I}})$, show that they form a conditionally projective family, and take their conditional projective limit using Theorem 2. This projective limit is used as a candidate for $k^{\mathrm{E}}$. To show that $k^{\mathrm{E}}$ indeed represents the posterior, we show that the two coincide on the cylinder sets (events which are preimages under projection of finite-dimensional events). From this, equality for all events follows by the Caratheodory theorem [1].
*Part (ii):* We only have to verify that the mappings $T^{\mathrm{I}}$ and probability kernels $k^{\mathrm{I}}$ indeed satisfy the definition of conjugacy, which is a straightforward computation. $\square$

# 5 Construction of Nonparametric Bayesian Models

Theorem 3(ii) states that conjugate models have conjugate marginals. Since, in the finite-dimensional case, conjugate Bayesian models are essentially limited to exponential families and

their natural conjugate priors[3], a consequence of the theorem is that we can only expect a non-parametric Bayesian model to be conjugate if it is constructed from exponential family marginals – assuming that the construction is based on a product space approach.

When an exponential family model and its conjugate prior are used in the construction, the form of the resulting model becomes generic: The posterior index $T$ of a conjugate exponential family Bayesian model is always given by the sufficient statistic $S$ in the form $T(x, (\alpha, y)) := (\alpha + 1, y + S(x))$. Addition commutes with projection, and hence the posterior indices $T^{\mathrm{I}}$ of a family of such models over all dimensions $I \in \mathcal{F}(E)$ satisfy the projection condition (4) if and only if the same condition is satisfied by the sufficient statistics $S^{\mathrm{I}}$ of the marginals. Accordingly, the infinite-dimensional posterior index $T^{\mathrm{E}}$ in Theorem 3 exists if and only if there is an infinite-dimensional "extension" $S^{\mathrm{E}}$ of the sufficient statistics $S^{\mathrm{I}}$ satisfying (4). If that is the case, $T^{\mathrm{E}}(x^{\mathrm{E}}, (\alpha, y^{\mathrm{E}})) := (\alpha + 1, y^{\mathrm{E}} + S^{\mathrm{E}}(x^{\mathrm{E}}))$ is a posterior index for the infinite-dimensional projective limit model. In the case of countable dimensions, Theorem 3 therefore implies a construction recipe for nonparametric Bayesian models from exponential family marginals; constructing the model boils down to checking whether the models selected as finite-dimensional marginals are conditionally projective, and whether the sufficient statistics satisfy the projection condition. An example construction, for a model on infinite permutations, is given in below. The following table summarizes some stochastic process models from the conjugate extension point of view:

| Marginals ($d$-dim) | Projective limit model | Observations (limit) |
|---|---|---|
| Bernoulli/Beta | Beta process; IBP | Binary arrays |
| Multin./Dirichlet | DP; CRP | Discrete distributions |
| Gaussian/Gaussian | GP/GP | (continuous) functions |
| Mallows/conjugate | Example below | Bijections $\mathbb{N} \to \mathbb{N}$ |

**A Construction Example.** The analysis of preference data, in which preferences are represented as permutations, has motivated the definition of distributions on permutations of an infinite number of items [9]. A finite permutation on $r$ items always implies a question such as "rank your favorite movies out of $r$ movies". A nonparametric approach can generalize the question to "rank your favorite movies". Meila and Bao [9] derived a model on infinite permutations, that is, on bijections of the set $\mathbb{N}$. We construct a nonparametric Bayesian model on bijections, with a likelihood component $P_{\mathrm{X}}^{\mathrm{E}}(X^{\mathrm{E}} | \Theta^{\mathrm{E}})$ equivalent to the model of Meila and Bao.

*Choice of marginals.* The finite-dimensional marginals are probability models of rankings of a finite number of items, introduced by Fligner and Verducci [6]. For permutations $\tau \in \mathbb{S}_r$ of length $r$, the model is defined by the exponential family density $p(\tau | \sigma, \theta) := Z(\theta)^{-1} \exp(\langle S(\tau \sigma^{-1}), \theta \rangle)$, where the sufficient statistic is the vector $S^r(\tau) := (S_1(\tau), \ldots, S_r(\tau))$ with components $S_j(\tau) := \sum_{l=j+1}^{r} \mathbb{I}\{\tau^{-1}(j) > \tau^{-1}(l)\}$. Roughly speaking, the model is a location-scale model, and the permutation $\sigma$ defines the distribution's mean. If all entries of $\theta$ are chosen identical as some constant, this constant acts as a concentration parameter, and the scalar product is equivalent to the Kendall metric on permutations. This metric measures distance between permutations as the minimum number of adjacent transpositions (i.e. swaps of neighboring entries) required to transform one permutation into the other. If the entries of $\theta$ differ, they can be regarded as weights specifying the relevance of each position in the ranking [6].

*Definition of marginals.* In the product space context, each finite set $I \in \mathcal{F}(E)$ of axis labels is a set of items to be permuted, and the marginal $P_{\Theta}^{\mathrm{I}}(\tau^{\mathrm{I}} | \sigma^{\mathrm{I}}, \theta^{\mathrm{I}})$ is a model on the corresponding finite permutation group $\mathbb{S}^{\mathrm{I}}$ on the elements of $I$. The sufficient statistics $S^{\mathrm{I}}$ maps each permutation to a vector of integers, and thus embeds the group $\mathbb{S}^{\mathrm{I}}$ into $\mathbb{R}^{\mathrm{I}}$. The mapping is one-to-one [6]. Projections, i.e. restrictions, on the group mean deletion of elements. A permutation $\tau^{\mathrm{J}}$ is restricted to a subset $I \subset J$ of indices by deleting all items indexed by $J \setminus I$, producing the restriction $\tau^{\mathrm{J}}|_{\mathrm{I}}$. We overload notation and write $\pi_{\mathrm{JI}}$ for both the restriction in the group $\mathbb{S}^{\mathrm{I}}$ and axes-parallel projection in the Euclidean space $\mathbb{R}^{\mathrm{I}}$, into which the sufficient statistic $S^{\mathrm{I}}$ embeds $\mathbb{S}^{\mathrm{I}}$. It follows from the definition of $S^{\mathrm{I}}$ that, whenever $\pi_{\mathrm{JI}}\tau^{\mathrm{J}} = \tau^{\mathrm{I}}$, then $\pi_{\mathrm{JI}}S^{\mathrm{J}}(\tau^{\mathrm{J}}) = S^{\mathrm{I}}(\tau^{\mathrm{I}})$. In other words, $\pi_{\mathrm{JI}} \circ S^{\mathrm{J}} = S^{\mathrm{I}} \circ \pi_{\mathrm{JI}}$, which is a stronger form of the projection condition $S^{\mathrm{J,-1}} \circ \pi_{\mathrm{JI}}^{-1} = \pi_{\mathrm{JI}}^{-1} \circ S^{\mathrm{I,-1}}$ given in Eq. 4. We will define a nonparametric Bayesian model that puts a prior on the infinite-dimensional analogue

of $\theta$, i.e. on the weight function $\theta^{\mathrm{E}}$. For $I \in \mathcal{F}(\mathbb{N})$, the marginal of the likelihood component is given by the density $p^{\mathrm{I}}(\tau^{\mathrm{I}}|\sigma^{\mathrm{I}},\theta^{\mathrm{I}}) := Z^{\mathrm{I}}(\theta^{\mathrm{I}})^{-1}\exp(\langle S^{\mathrm{I}}(\tau^{\mathrm{I}}(\sigma^{\mathrm{I}})^{-1}),\theta^{\mathrm{I}}\rangle)$. The corresponding natural conjugate prior on $\theta^{\mathrm{I}}$ has density $q^{\mathrm{I}}(\theta^{\mathrm{I}}|\alpha,y^{\mathrm{I}}) \propto \exp(\langle\theta^{\mathrm{I}},y^{\mathrm{I}}\rangle - \alpha\log Z^{\mathrm{I}}(\theta^{\mathrm{I}}))$. Since the model is an exponential family model, the posterior index is of the form $T^{\mathrm{I}}((\alpha,y^{\mathrm{I}}),\tau^{\mathrm{I}}) = (\alpha+1, y^{\mathrm{I}}+S^{\mathrm{I}}(\tau^{\mathrm{I}}))$, and since $S^{\mathrm{I}}$ is projective in the sense of Eq. 4, so is $T^{\mathrm{I}}$. The prior and likelihood densities above define two families $P^{\mathrm{I}}(X^{\mathrm{I}}|\Theta^{\mathrm{I}})$ and $P^{\mathrm{I}}(\Theta^{\mathrm{I}}|\Psi)$ of measures over all finite dimensions $I \in \mathcal{F}(E)$. It is reasonably straightforward to show that both families are conditionally projective, and so is the family of the corresponding posteriors. Each therefore has a projective limit, and the projective limit of the posteriors is the posterior of the projective limit $P^{\mathrm{E}}(X^{\mathrm{E}}|\Theta^{\mathrm{E}})$ under prior $P^{\mathrm{E}}(\Theta^{\mathrm{E}})$.

*Posterior index.* The posterior index of the infinite-dimensional model can be derived by means of Theorem 3: To get rid of the hyperparameters, we first fix a value $\psi^{\mathrm{E}} := (\alpha,y^{\mathrm{E}})$ of the infinite-dimensional hyperparameter, and only consider the corresponding infinite-dimensional prior $P^{\mathrm{E}}_\Theta(\Theta^{\mathrm{E}}|\Psi^{\mathrm{E}}=\psi^{\mathrm{E}})$, with its marginals $P^{\mathrm{I}}_\Theta(\Theta^{\mathrm{I}}|\Psi^{\mathrm{I}}=\pi_{\mathrm{EI}}\psi^{\mathrm{E}})$. Now define a function $S^{\mathrm{E}}$ on the bijections of $\mathbb{N}$ as follows. For each bijection $\tau : \mathbb{N} \to \mathbb{N}$, and each $j \in \mathbb{N}$, set $S^{\mathrm{E}}_j(\tau) := \sum_{l=j+1}^{\infty}\mathbb{I}\{\tau^{-1}(j) > \tau^{-1}(l)\}$. Since $\tau^{-1}(j)$ is a finite number for any $j \in \mathbb{N}$, the indicator function is non-zero only for a finite number of indices $l$, such that the entries of $S^{\mathrm{E}}$ are always finite. Then $S^{\mathrm{E}}$ satisfies the projection condition $S^{\mathrm{E},\text{-}1} \circ \pi_{\mathrm{EI}}^{\text{-}1} = \pi_{\mathrm{EI}}^{\text{-}1}S^{\mathrm{I},\text{-}1}$ for all $I \in \mathcal{F}(E)$. As candidate posterior index, we define the function $T^{\mathrm{E}}((\alpha,y^{\mathrm{E}}),\tau^{\mathrm{E}}) = (\alpha+1, y^{\mathrm{E}}+S^{\mathrm{E}}(\tau^{\mathrm{E}}))$ for $y^{\mathrm{E}} \in \Omega^{\mathbb{N}}_\theta$. Then $T^{\mathrm{E}}$ also satisfies the projection condition (4) for any $I \in \mathcal{F}(E)$. By Theorem 3, this makes $T^{\mathrm{E}}$ a posterior index for the projective limit model.

# 6 Discussion and Conclusion

We have shown how nonparametric Bayesian models can be constructed from finite-dimensional Bayes equations, and how conjugacy properties of the finite-dimensional models carry over to the infinite-dimensional, nonparametric case. We also have argued that conjugate nonparametric Bayesian models arise from exponential families.

A number of interesting questions could not be addressed within the scope of this paper, including (1) the extension to model properties other than conjugacy and (2) the generalization to uncountable dimensions. For example, a model property which is closely related to conjugacy is sufficiency [14]. In this case, we would ask whether the existence of sufficient statistics for the finite-dimensional marginals implies the existence of a sufficient statistic for the nonparametric Bayesian model, and whether the infinite-dimensional sufficient statistic can be explicitly constructed. Second, the results presented here are restricted to the case of countable dimensions. This restriction is inconvenient, since the natural product space representations of, for example, Gaussian and Dirichlet processes on the real line have uncountable dimensions. The GP (on continuous functions) and the DP are within the scope of our results, as both can be derived by means of countable-dimensional surrogate constructions: Since continuous functions on $\mathbb{R}$ are completely determined by their values on $\mathbb{Q}$, a GP can be constructed on the countable-dimensional product space $\mathbb{R}^{\mathbb{Q}}$. Analogous constructions have been proposed for the DP [7]. The drawback of this approach is that the actual random draw is just a partial version of the object of interest, and formally has to be completed e.g. into a continuous function or a probability measure *after* it is sampled. On the other hand, uncountable product space constructions are subject to all the subtleties of stochastic process theory, many of which do not occur in countable dimensions. The application of construction methods to conditional probabilities also becomes more complicated (roughly speaking, the point-wise application of the Kolmogorov theorem in the proof of Theorem 2 is not possible if the dimension is uncountable).

Product space constructions are by far not the only way to define nonparametric Bayesian models. A Pólya tree model [7], for example, is much more intuitive to construct by means of a binary partition argument than from marginals in product space. As far as characterization results, such as which models can be conjugate, are concerned, our results are still applicable, since the set of Polyá trees can be embedded into a product space. However, the marginals may then not be the marginals in terms of which we "naturally" think about the model. Nonetheless, we have hopefully demonstrated that the theoretical results are applicable for the construction of an interesting and practical range of nonparametric Bayesian models.

**Acknowledgments.** I am grateful to Joachim M. Buhmann, Zoubin Ghaharamani, Finale Doshi-Velez and the reviewers for helpful comments. This work was in part supported by EPSRC grant EP/F028628/1.

## Footnotes

[1]This problem is unfortunately often neglected in the statistics literature, and measures in uncountable dimensions are "constructed" by means of the extension theorem (such as in the original paper [5] on the Dirichlet process). See e.g. [1] for theoretical background, and [7] for a rigorous construction of the DP.

[2] Complete proofs for both theorems in this paper are provided as supplementary material.

[3]Mixtures of conjugate priors are conjugate in the sense of closure under sampling [4], but the posterior index in Def. 2 has to be evaluated for each mixture component individually. An example of a conjugate model *not* in the exponential family is the uniform distribution on $[0, \theta]$ with a Pareto prior [12].

## References

[1] H. Bauer. *Probability Theory*. W. de Gruyter, 1996.

[2] M. J. Beal, Z. Ghahramani, and C. E. Rasmussen. The infinite hidden Markov model. In *Advances in Neural Information Processing Systems*, 2001.

[3] P. Billingsley. Probability and measure, 1995.

[4] S. R. Dalal and W. J. Hall. Approximating priors by mixtures of natural conjugate priors. *Annals of Statistics*, 45(2):278–286, 1983.

[5] T. S. Ferguson. A Bayesian analysis of some nonparametric problems. *Annals of Statistics*, 1(2), 1973.

[6] M. A. Fligner and J. S. Verducci. Distance based ranking models. *Journal of the Royal Statistical Society B*, 48(3):359–369, 1986.

[7] J. K. Ghosh and R. V. Ramamoorthi. *Bayesian Nonparametrics*. Springer, 2002.

[8] T. L. Griffiths and Z. Ghahramani. Infinite latent feature models and the Indian buffet process. In *Advances in Neural Information Processing Systems*, 2005.

[9] M. Meilă and L. Bao. Estimation and clustering with infinite rankings. In *Uncertainty in Artificial Intelligence*, 2008.

[10] M. M. Rao. *Conditional Measures and Applications*. Chapman & Hall, second edition, 2005.

[11] C. E. Rasmussen and C. K. I. Williams. *Gaussian Processes for Machine Learning*. MIT Press, 2006.

[12] C. P. Robert. *The Bayesian Choice*. Springer, 1994.

[13] D. M. Roy and Y. W. Teh. The Mondrian process. In *Advances in Neural Information Processing Systems*, 2009.

[14] M. J. Schervish. *Theory of Statistics*. Springer, 1995.

[15] Y. W. Teh, M. I. Jordan, M. J. Beal, and D. M. Blei. Hierarchical Dirichlet processes. *Journal of the American Statistical Association*, (476):1566–1581, 2006.

[16] S. G. Walker, P. Damien, P. W. Laud, and A. F. M. Smith. Bayesian nonparametric inference for random distributions and related functions. *Journal of the Royal Statistical Society B*, 61(3):485–527, 1999.

[17] L. Wasserman. *All of Nonparametric Statistics*. Springer, 2006.

